# Connectionist Optimisation of Tied Mixture Hidden Markov Models

Steve Renals
Nelson Morgan
ICSI
Berkeley CA 94704
USA

Hervé Bourlard

L&H Speechproducts
Ieper B-9800
Belgium

Horacio Franco
Michael Cohen
SRI International
Menlo Park CA 94025
USA

## Abstract

Issues relating to the estimation of hidden Markov model (HMM) local probabilities are discussed. In particular we note the isomorphism of radial basis functions (RBF) networks to tied mixture density modelling; additionally we highlight the differences between these methods arising from the different training criteria employed. We present a method in which connectionist training can be modified to resolve these differences and discuss some preliminary experiments. Finally, we discuss some outstanding problems with discriminative training.

## 1 INTRODUCTION

In a statistical approach to continuous speech recognition the desired quantity is the posterior probability $P(\mathbf{W}_1^W | \mathbf{X}_1^T, \Theta)$ of a word sequence $\mathbf{W}_1^W = \mathbf{w}_1, ..., \mathbf{w}_W$ given the acoustic evidence $\mathbf{X}_1^T = \mathbf{x}_1, ..., \mathbf{x}_T$ and the parameters of the speech model used $\Theta$. Typically a set of models is used, to separately model different units of speech. This probability may be re-expressed using Bayes' rule:

$$
(1) \qquad P(\mathbf{W}_1^W | \mathbf{X}_1^T, \Theta) = \frac{P(\mathbf{X}_1^T | \mathbf{W}_1^W, \Theta) P(\mathbf{W}_1^W | \Theta)}{P(\mathbf{X}_1^T | \Theta)}
$$

$$
= \frac{P(\mathbf{X}_1^T | \mathbf{W}_1^W, \Theta) P(\mathbf{W}_1^W | \Theta)}{\sum_{\mathbf{W}'} P(\mathbf{X}_1^T | \mathbf{W}', \Theta) P(\mathbf{W}' | \Theta)}.
$$

$P(\mathbf{X}_1^T | \mathbf{W}_1^W, \Theta) / P(\mathbf{X}_1^T | \Theta)$ is the acoustic model. This is the ratio of the likelihood of the acoustic evidence given the sequence of word models, to the probability of the acoustic data being generated by the complete set of models. $P(\mathbf{X}_1^T|\Theta)$ may be regarded as a normalising term that is constant (across models) at recognition time. However at training time the parameters $\Theta$ are being adapted, thus $P(\mathbf{X}_1^T|\Theta)$ is no longer constant. The prior, $P(\mathbf{W}_1^W|\Theta)$, is obtained from a language model.

The basic unit of speech, typically smaller than a word (here we use phones), is modelled by a hidden Markov model (HMM). Word models consist of concatenations of phone HMMs (constrained by pronunciations stored in a lexicon), and sentence models consist of concatenations of word HMMs (constrained by a grammar). The lexicon and grammar together make up a language model, specifying prior probabilities for sentences, words and phones.

A HMM is a stochastic automaton defined by a set of states $q_i$, a topology specifying allowed state transitions and a set of local probability density functions (PDFs) $P(\mathbf{x}_t, q_i|q_j, \mathbf{X}_1^{t-1})$. Making the further assumptions that the output at time $t$ is independent of previous outputs and depends only on the current state, we may separate the local probabilities into state transition probabilities $P(q_i|q_j)$ and output PDFs $P(\mathbf{x}_t|q_i)$. A set of initial state probabilities must also be specified.

The parameters of a HMM are usually set via a maximum likelihood procedure that optimally estimates the joint density $P(q, \mathbf{x}|\Theta)$. The forward-backward algorithm, a provably convergent algorithm for this task, is extremely efficient in practice. However, in speech recognition we do not wish to make the best model of the data $\{\mathbf{x}, \mathbf{q}\}$ given the model parameters; we want to make the optimal discrimination between classes at each time. This can be better achieved by computing a discriminant $P(q|\mathbf{x}, \Theta)$. Note that in this case we do not model the input density $P(\mathbf{x}|\Theta)$.

We may estimate $P(q|\mathbf{x}, \Theta)$ using a feed-forward network trained to an entropy criterion (Bourlard & Wellekens, 1989). However, we require likelihoods of the form $P(\mathbf{x}|q, \Theta)$, as HMM output probabilities. We may convert posterior probabilities to scaled likelihoods $P(\mathbf{x}|q, \Theta)/P(\mathbf{x}|\Theta)$, by dividing the network outputs by the relative frequencies of each class[1]. Note that we are not using connectionist training to obtain density estimates here; we are obtaining a ratio and not modelling $P(\mathbf{x}|\Theta)$. This ratio is the quantity that we wish to maximise: this corresponds to maximising $P(\mathbf{x}|q_c, \Theta)$ and minimising $P(\mathbf{x}|q_i, \Theta)$, $i \neq c$, where $q_c$ is the correct class. We have used discriminatively trained networks to estimate the output PDFs (Bourlard & Morgan, 1991; Renals et al., 1991, 1992), and have obtained superior results to maximum likelihood training on continuous speech recognition tasks.

In this paper, we are mainly concerned with radial basis function (RBF) networks. A RBF network generally has a single hidden layer, whose units may be regarded as computing local (or approximately local) densities, rather than global decision surfaces. The resultant posteriors are obtained by output units that combine these local densities. We are interested in using RBF networks for various reasons:

- A RBF network is isomorphic to a tied mixture density model, although the training criterion is typically different. The relationship between the two is explored in this paper.

- The locality of RBFs makes them suitable for situations in which the input

distribution may change (e.g. speaker adaptation). Surplus RBFs in a region of the input space where data no longer occurs will not effect the final classification. This is not so for sigmoidal hidden units in a multi-layer perceptron (MLP), which have a global effect.

- RBFs are potentially more computationally efficient than MLPs at both training and recognition time.

## 2   TIED MIXTURE HMM

Tied mixtures of Gaussians have proven to be powerful PDF estimators in HMM speech recognition systems (Huang & Jack, 1989; Bellegarda & Nahamoo, 1990). The resulting systems are also known as semi-continuous HMMs. Tied mixture density estimation may be regarded as an interpolation between discrete and continuous density modelling Essentially, tied mixture modelling has a single "codebook" of Gaussians shared by all output PDFs. Each of these PDFs has its own set of mixture coefficients used to combine the individual Gaussians. If $f_k(\mathbf{x}|q_k)$ is the output PDF of state $q_k$, and $N_j(\mathbf{x}|\mu_j, \Sigma_j)$ are the component Gaussians, then:

$$(2) \qquad f_k(\mathbf{x}|q_k, \Theta) = \sum_j a_{kj} N_j(\mathbf{x}|\mu_j, \Sigma_j)$$

$$\sum_j a_{kj} = 1 \qquad 0 \le a_{kj} \le 1,$$

where $a_{kj}$ is an element of the matrix of mixture coefficients (which may be interpreted as the prior probability $P(\mu_j, \Sigma_j|q_k)$) defining how much component density $N_j(\mathbf{x}|\mu_j, \Sigma_j)$ contributes to output PDF $f_k(\mathbf{x}|q_k, \Theta)$. Alternatively this may be regarded as "fuzzy" vector quantisation.

## 3   RADIAL BASIS FUNCTIONS

The radial basis functions (RBF) network was originally introduced as a means of function interpolation (Powell, 1985; Broomhead & Lowe, 1988). A set of $K$ approximating functions, $f_k(\mathbf{x})$ is constructed from a set of $J$ basis functions $\phi(\mathbf{x})$:

$$(3) \qquad f_k(\mathbf{x}) = \sum_{j=1}^{J} a_{kj}\phi_j(\mathbf{x}) \qquad 1 \le k \le K$$

This equation defines a RBF network with $J$ RBFs (hidden units) and $K$ outputs. The output units here are linear, with weights $a_{kj}$. The RBFs are typically Gaussians, with means $\mu_j$ and covariance matrices $\Sigma_j$:

$$(4) \qquad \phi_j(\mathbf{x}) = R \exp\left(-\frac{1}{2}(\mathbf{x} - \mu_j)^T \Sigma_j^{-1}(\mathbf{x} - \mu_j)\right),$$

where $R$ is a normalising constant. The covariance matrix is frequently assumed to be diagonal[2].

Such a network has been used for HMM output probability estimation in continuous speech recognition (Renals et al., 1991) and an isomorphism to tied-mixture HMMs was noted. However, there is a mismatch between the posterior probabilities estimated by the network and the likelihoods required for the HMM decoding. Previously this was resolved by dividing the outputs by the relative frequencies of each state. It would be desirable, though, to retain the isomorphism to tied mixtures: specifically we wish to interpret the hidden-to-output weights of an RBF network as the mixture coefficients of a tied mixture likelihood function. This can be achieved by defining the transfer units of the output units to implement Bayes' rule, which relates the posterior $g_k(\mathbf{x})$ to the likelihood $f_k(\mathbf{x})$:

$$(5) \qquad g_k(\mathbf{x}) = \frac{f_k(\mathbf{x})P(q_k)}{\sum_{l=1}^{K} f_l(\mathbf{x})P(q_l)}.$$

Such a transfer function ensures the output units sum to 1; if $f_k(\mathbf{x})$ is guaranteed non-negative, then the outputs are formally probabilities. The output of such a network is a probability distribution and we are using '1-from-K' training: thus the relative entropy $E$ is simply:

$$(6) \qquad E = -\log g_c(\mathbf{x}),$$

where $q_c$ is the desired output class (HMM distribution). Bridle (1990) has demonstrated that minimising this error function is equivalent to maximising the mutual information between the acoustic evidence and HMM state sequence.

If we wish to interpret the weights as mixture coefficients, then we must ensure that they are non-negative and sum to 1. This may be achieved using a normalised exponential (softmax) transformation:

$$(7) \qquad a_{kj} = \frac{\exp(w_{kj})}{\sum_h \exp(w_{kh})}.$$

The mixture coefficients $a_{kj}$ are used to compute the likelihood estimates, but it is the derived variables $w_{kj}$ that are used in the unconstrained optimisation.

## 3.1 TRAINING

Steepest descent training specifies that:

$$(8) \qquad \frac{\partial w_{kj}}{\partial t} = -\frac{\partial E}{\partial w_{kj}}.$$

Here $E$ is the relative entropy objective function (6). We may decompose the right hand side of this by a careful application of the chain rule of differentiation:

$$(9) \qquad \frac{\partial E}{\partial w_{kj}} = \sum_{l=1}^{K} \frac{\partial E}{\partial g_l(\mathbf{x})} \frac{\partial g_l(\mathbf{x})}{\partial f_k(\mathbf{x})} \sum_{h=1}^{J} \frac{\partial f_k(\mathbf{x})}{\partial a_{kh}} \frac{\partial a_{kh}}{\partial w_{kj}}.$$

We may write down expressions for each of these partials (where $\delta_{ab}$ is the Kronecker delta and $q_c$ is the desired state):

$$(10) \qquad \frac{\partial E}{\partial g_l(\mathbf{x})} = -\frac{\delta_{cl}}{g_c}$$

$$(11) \qquad \frac{\partial g_l(\mathbf{x})}{\partial f_k(\mathbf{x})} = \frac{g_k(\mathbf{x})}{f_k(\mathbf{x})}(\delta_{kl} - g_l)$$

$$(12) \qquad \frac{\partial f_k(\mathbf{x})}{\partial a_{kh}} = \phi_h(\mathbf{x})$$

$$(13) \qquad \frac{\partial a_{kh}}{\partial w_{kj}} = a_{kh}(\delta_{hj} - a_{kj}).$$

Substituting (10), (11), (12) and (13) into (9) we obtain:

$$(14) \qquad \frac{\partial E}{\partial w_{kj}} = \frac{1}{f_k(\mathbf{x})}(g_k(\mathbf{x}) - \delta_{kc})\, a_{kj}\left(\phi_j(\mathbf{x}) - f_k(\mathbf{x})\right).$$

Apart from the added terms due to the normalisation of the weights, the major difference in the gradient compared with using a sigmoid or softmax transfer function is the $1/f_k(\mathbf{x})$ factor. To some extent we may regard this as a dimensional term.

The required gradient is simpler if we construct the network to estimate log likelihoods, replacing $f_k(\mathbf{x})$ with $z_k(\mathbf{x}) = \log f_k(\mathbf{x})$:

$$(15) \qquad z_k(\mathbf{x}) = \sum_j w_{kj}\phi_j(\mathbf{x})$$

$$(16) \qquad g_k(\mathbf{x}) = \frac{p(q_k)\exp(z_k(\mathbf{x}))}{\sum_l p(q_l)\exp(z_l(\mathbf{x}))}.$$

Since this is in the log domain, no constraints on the weights are required. The new gradient we need is:

$$(17) \qquad \frac{\partial g_l(\mathbf{x})}{\partial f_k(\mathbf{x})} = g_k(\delta_{kl} - g_l).$$

Thus the gradient of the error is:

$$(18) \qquad \frac{\partial E}{\partial w_{kj}} = (g_k(\mathbf{x}) - \delta_{ck})\phi_j(\mathbf{x}).$$

Since we are in log domain, the $1/f_k(\mathbf{x})$ factor is additive and thus disappears from the gradient. This network is similar to Bridle's softmax, except here uniform priors are not assumed; the gradient is of identical form, though. In this case the weights do not have a simple relationship with the mixture coefficients obtained in tied mixture density modelling.

We may also train the means and variances of the RBFs by back-propagation of error; the gradients are straightforward.

## 3.2 PRELIMINARY EXPERIMENTS

We have experimented with both the Bayes' rule transfer function (5) and the variant in the log domain (16). We used a phoneme classification task, with a

database consisting of 160,000 frames of continuous speech. We typically computed the parameters of the RBFs by a k-means clustering process. We found that the gradient resulting from the first transfer function (14) had a tendency to numerical instability, due to the $1/f$ term; thus most of our experiments have used the log domain transfer function.

In experiments using a 1000 RBFs, we have obtained frame classification rates of 52%. This is somewhat poorer than the frame classification we obtain using a 512 hidden unit MLP (59%). We are investigating improvements to our procedure, including variations to the learning schedule, the use of the EM algorithm to set RBF parameters and the use of priors on the weight matrix.

## 4    PROBLEMS WITH DISCRIMINATIVE TRAINING

### 4.1    UNLABELLED DATA

A problem arises from the use of unlabelled or partially labelled data. When training a speech recogniser, we typically know the word sequence for an utterance, but we do not have a time-aligned phonetic transcription. This is a case of partially labelled data: a training set of data pairs $\{x_t, q_t\}$ is unavailable, but we do not have purely unlabelled data $\{x_t\}$. Instead, we have the constraining information of the word sequence $\mathbf{W}$. Thus $P(q_i|x_t)$ may be decomposed as:

$$(19) \qquad P(q_i|x_t) = P(q_i|x_t, \mathbf{W})P(\mathbf{W}|x_t) .$$

We usually make the further approximation that the optimal state sequence is much more likely than any competing state sequence. Thus, $P(q_c|x_t) = 1$, and the probabilities of all other states at time $t$ are 0. This most likely state sequence (which may be computed using a forced Viterbi alignment) is often used as the desired outputs for a discriminatively trained network. Using this alignment implicitly assumes model correctness; however, we use discriminative training because we believe the HMMs are an inadequate speech model. Hence there is a mismatch between the maximum likelihood labelling and alignment, and the discriminative training used for the networks.

It may be that this mismatch is responsible for the lack of robustness of discriminative training (compared with pure maximum likelihood training) in vocabulary independent speech recognition tasks (Paul et al., 1991). The assumption of model correctness used to generate the labels may have the effect of further embedding specifics of the training data into the final models. A solution to this problem may be to use a probabilistic alignment, with a distribution over labels at each timestep. This could be computed using the forward-backward algorithm, rather than the Viterbi approximation. This maximum likelihood approach still assumes model correctness of course. A discriminative approach to this problem would also attempt to infer distributions over labels. A basic goal might be to sharpen the distribution toward the maximum likelihood estimate. An example of such a method is the 'phantom targets' algorithm introduced by Bridle & Cox (1991).

These optimisations are local: the error is not propagated through time. Algorithms for globally optimising discriminative training have been proposed (e.g. Bengio et al., these proceedings), but are not without problems, when used with a constrain-

ing language model. The problem is that to compute the posterior, the ratio of the probabilities of generating the correct utterance and generating all allowable utterances must be computed.

## 4.2  THE PRIORS

It has been shown, both theoretically and in practice, that the training and recognition procedures used with standard HMMs remain valid for posterior probabilities (Bourlard & Wellekens, 1989). Why then do we replace these posterior probabilities with likelihoods?

The answer to this problem lies in a mismatch between the prior probabilities given by the training data and those imposed by the topology of the HMMs. Choosing the HMM topology also amounts to fixing the priors. For instance, if classes $q_k$ represent phones, prior probabilities $P(q_k)$ are fixed when word models are defined as particular sequences of phone models. This discussion can be extended to different levels of processing: if $q_k$ represents sub-phonemic states and recognition is constrained by a language model, prior probabilities $q_k$ are fixed by (and can be calculated from) the phone models, word models and the language model. Ideally, the topologies of these models would be inferred directly from the training data, by using a discriminative criterion which implicitly contains the priors. Here, at least in theory, it would be possible to start from fully-connected models and to determine their topology according to the priors observed on the training data. Unfortunately this results in a huge number of parameters that would require an unrealistic amount of training data to estimate them significantly. This problem has also been raised in the context of language modelling (Paul et al., 1991).

Since the ideal theoretical solution is not accessible in practice, it is usually better to dispose of the poor estimate of the priors obtained using the training data, replacing them with "prior" phonological or syntactic knowledge.

## 5  CONCLUSION

Having discussed the similarities and differences between RBF networks and tied mixture density estimators, we present a method that attempts to resolve a mismatch between discriminative training and density estimation. Some preliminary experiments relating to this approach were discussed; we are currently performing further speech recognition experiments using these methods. Finally we raised some important issues pertaining to discriminative training.

### Acknowledgement

This work was partially funded by DARPA contract MDA904-90-C-5253.

## Footnotes

[1]These are the estimates of $P(q_i)$ implicitly used during classifier training.

[2]This is often reasonable for speech applications, since mel or PLP cepstral coefficients are orthogonal.

### References

Bellegarda, J. R. & Nahamoo, D. (1990). Tied mixture continuous parameter modeling for speech recognition. *IEEE Transactions on Acoustics, Speech and Signal Processing, 38*, 2033–2045.

Bourlard, H. & Morgan, N. (1991). Conectionist approaches to the use of Markov models for continuous speech recognition. In Lippmann, R. P., Moody, J. E., & Touretzky, D. S. (Eds.), *Advances in Neural Information Processing Systems*, Vol. 3, pp. 213–219. Morgan Kaufmann, San Mateo CA.

Bourlard, H. & Wellekens, C. J. (1989). Links between Markov models and multi-layer perceptrons. In Touretzky, D. S. (Ed.), *Advances in Neural Information Processing Systems*, Vol. 1, pp. 502–510. Morgan Kaufmann, San Mateo CA.

Bridle, J. S. & Cox, S. J. (1991). RecNorm: Simultaneous normalisation and classification applied to speech recognition. In Lippmann, R. P., Moody, J. E., & Touretzky, D. S. (Eds.), *Advances in Neural Information Processing Systems*, Vol. 3, pp. 234–240. Morgan Kaufmann, San Mateo CA.

Bridle, J. S. (1990). Training stochastic model recognition algorithms as networks can lead to maximum mutual information estimation of parameters. In Touretzky, D. S. (Ed.), *Advances in Neural Information Processing Systems*, Vol. 2, pp. 211–217. Morgan Kaufmann, San Mateo CA.

Broomhead, D. S. & Lowe, D. (1988). Multi-variable functional interpolation and adaptive networks. *Complex Systems, 2*, 321–355.

Huang, X. D. & Jack, M. A. (1989). Semi-continuous hidden Markov models for speech signals. *Computer Speech and Language, 3*, 239–251.

Paul, D. B., Baker, J. K., & Baker, J. M. (1991). On the interaction between true source, training and testing language models. In *Proceedings IEEE International Conference on Acoustics, Speech and Signal Processing*, pp. 569–572 Toronto.

Powell, M. J. D. (1985). Radial basis functions for multi-variable interpolation: a review. Tech. rep. DAMPT/NA12, Dept. of Applied Mathematics and Theoretical Physics, University of Cambridge.

Renals, S., McKelvie, D., & McInnes, F. (1991). A comparative study of continuous speech recognition using neural networks and hidden Markov models. In *Proceedings IEEE International Conference on Acoustics, Speech and Signal Processing*, pp. 369–372 Toronto.

Renals, S., Morgan, N., Cohen, M., & Franco, H. (1992). Connectionist probability estimation in the DECIPHER speech recognition system. In *Proceedings IEEE International Conference on Acoustics, Speech and Signal Processing* San Francisco. In press.